# Fast and Scalable Training of Semi-Supervised CRFs with Application to Activity Recognition

**Maryam Mahdaviani**
Computer Science Department
University of British Columbia
Vancouver, BC, Canada

**Tanzeem Choudhury**
Intel Research
1100 NE 45th Street
Seattle, WA 98105,USA

## Abstract

We present a new and efficient semi-supervised training method for parameter estimation and feature selection in conditional random fields (CRFs). In real-world applications such as activity recognition, unlabeled sensor traces are relatively easy to obtain whereas labeled examples are expensive and tedious to collect. Furthermore, the ability to automatically select a small subset of discriminatory features from a large pool can be advantageous in terms of computational speed as well as accuracy. In this paper, we introduce the semi-supervised virtual evidence boosting (sVEB) algorithm for training CRFs – a semi-supervised extension to the recently developed virtual evidence boosting (VEB) method for feature selection and parameter learning. The objective function of sVEB combines the unlabeled conditional entropy with labeled conditional pseudo-likelihood. It reduces the overall system cost as well as the human labeling cost required during training, which are both important considerations in building real-world inference systems. Experiments on synthetic data and real activity traces collected from wearable sensors, illustrate that sVEB benefits from both the use of unlabeled data and automatic feature selection, and outperforms other semi-supervised approaches.

## 1 Introduction

Conditional random fields (CRFs) are undirected graphical models that have been successfully applied to the classification of relational and temporal data [1]. Training complex CRF models with large numbers of input features is slow, and exact inference is often intractable. The ability to select the most informative features as needed can reduce the training time and the risk of over-fitting of parameters. Furthermore, in complex modeling tasks, obtaining the large amount of labeled data necessary for training can be impractical. On the other hand, large unlabeled datasets are often easy to obtain, making semi-supervised learning methods appealing in various real-world applications.

The goal of our work is to build an activity recognition system that is not only accurate but also scalable, efficient, and easy to train and deploy. An important application domain for activity recognition technologies is in health-care, especially in supporting elder care, managing cognitive disabilities, and monitoring long-term health. Activity recognition systems will also be useful in smart environments, surveillance, emergency and military missions. Some of the key challenges faced by current activity inference systems are the amount of human effort spent in labeling and feature engineering and the computational complexity and cost associated with training. Data labeling also has privacy implications because it often requires human observers or recording of video. In this paper, we introduce a fast and scalable semi-supervised training algorithm for CRFs and evaluate its classification performance on extensive real world activity traces gathered using wearable sensors. In addition to being computationally efficient, our proposed method reduces the amount of labeling required during training, which makes it appealing for use in real world applications.

Several supervised techniques have been proposed for feature selection in CRFs. For discrete features, McCallum [2] suggested an efficient method for feature induction by iteratively increasing conditional log-likelihood. Dietterich [3] applied gradient tree boosting to select features in CRFs by combining boosting with parameter estimation for 1D linear-chain models. Boosted random fields (BRFs) [4] combine boosting and belief propagation for feature selection and parameter estimation for densely connected graphs that have weak pairwise connections. Recently, Liao *et.al.* [5] developed a more general version of BRFs, called virtual evidence boosting (VEB) that does not make any assumptions about graph connectivity or the strength of pairwise connections. The objective function in VEB is a soft version of maximum pseudo-likelihood (MPL), where the goal is to maximize the sum of local log-likelihoods given soft evidence from its neighbors. This objective function is similar to that used in boosting, which makes it suitable for unified feature selection and parameter estimation. This approximation applies to any CRF structures and leads to a significant reduction in training complexity and time. Semi-supervised training techniques have been extensively explored in the case of generative models and naturally fit under the expectation maximization framework [6]. However, it is not straight forward to incorporate unlabeled data in discriminative models using the traditional conditional likelihood criteria. A few semi-supervised training methods for CRFs have been proposed that introduce dependencies between nearby data points [7, 8]. More recently, Grandvalet and Bengio [9] proposed a minimum entropy regularization framework for incorporating unlabeled data. Jiao *et.al.* [10] used this framework and proposed an objective function that combines the conditional likelihood of the labeled data with the conditional entropy of the unlabeled data to train 1D CRFs, which was extended to 2D lattice structures by Lee *et.al.* [11].

In our work, we combine the minimum entropy regularization framework for incorporating unlabeled data with VEB for training CRFs. The contributions of our work are: (i) semi-supervised virtual evidence boosting (sVEB) - an efficient technique for simultaneous feature selection and semi-supervised training of CRFs, which to the best of our knowledge is the first method of its kind, (ii) experimental results that demonstrate the strength of sVEB, which consistently outperforms other training techniques on synthetic data and real-world activity classification tasks, and (iii) analysis of the time and complexity requirements of our algorithm, and comparison with other existing techniques that highlight the significant computational advantages of our approach. The sVEB algorithm is fast and easy to implement and has the potential of being broadly applicable.

## 2   Approaches to training of Conditional Random Fields

Maximum likelihood parameter estimation in CRFs involves maximizing the overall conditional log-likelihood, where $\mathbf{x}$ is the observation sequence and $\mathbf{y}$ is the hidden state sequence:

$$L(\theta) = \log(p(\mathbf{y}|\mathbf{x}, \theta)) - \|\theta\|/2 = \log \frac{\exp(\sum\limits_{k=1}^{K} \theta_k f_k(\mathbf{x}, \mathbf{y}))}{\sum\limits_{\mathbf{y}'} \exp(\sum\limits_{k=1}^{K} \theta_k f_k(\mathbf{x}, \mathbf{y}'))} - \|\theta\|/2 \qquad (1)$$

The conditional distribution is defined by a log-linear combination of $k$ features functions $f_k$ associated with weight $\theta_k$. A regularizer on $\theta$ is used to keep the weights from getting too large and to avoid overfitting[1]. For large CRFs exact inference is often intractable and approximate methods such as mean field approximation or loopy belief propagation [12, 13] are used.

An alternative to approximating the conditional likelihood is to change the objective function. MPL [14] and VEB [5] are such techniques. For MPL the CRF is cut into a set of independent patches; each patch consists of a hidden node or class label $y_i$, the true value of its direct neighbors and the observations, i.e., the Markov Blanket($MB_{y_i}$) of the node. The parameter estimation then becomes maximizing the pseudo log-likelihood:

$$L_{pseudo}(\theta) = \sum_{i=1}^{N} \log(p(y_i|MB_{y_i}, \theta)) = \sum_{i=1}^{N} \log \frac{\exp(\sum\limits_{k=1}^{K} \theta_k f_k(MB_{y_i}, y_i))}{\sum\limits_{y_i'} \exp(\sum\limits_{k=1}^{K} \theta_k f_k(MB_{y_i'}, y_i'))}$$

MPL has been known to over-estimate the dependency parameters in some cases and there is no general guideline on when it can be safely used [15].

## 2.1 Virtual evidence boosting

By extending the standard LogitBoost algorithm [16], VEB integrates boosting based feature selection into CRF training. The objective function used in VEB is very similar to MPL, except that VEB uses the messages from the neighboring nodes as virtual evidence instead of using the true labels of neighbors. The use of virtual evidence helps to reduce over-estimation of neighborhood dependencies. We briefly explain the approach here but please refer to [5] for more detail.

VEB incorporates two types of observations nodes: (i) *hard* evidence corresponding to the observations $\mathbf{ve}(\mathbf{x}_i)$, which are indicator functions at the observation values and (ii) *soft* evidence, corresponding to the messages from neighboring nodes $\mathbf{ve}(\mathbf{n}(y_i))$, which are discrete distributions over the hidden states. Let $\mathbf{ve_i} \triangleq \{\mathbf{ve}(\mathbf{x}_i), \mathbf{ve}(n(y_i))\}$. The objective function of VEB is as follows:

$$L_{VEB}(\theta) = \sum_{i=1}^{N} \log(p(y_i|\mathbf{ve_i}, \theta)), \ where \ p(y_i|\mathbf{ve_i}, \theta) = \frac{\sum\limits_{\mathbf{ve_i}} \mathbf{ve_i} \exp(\sum\limits_{k=1}^{K} \theta_k f_k(\mathbf{ve_i}, y_i))}{\sum\limits_{y_i'} \sum\limits_{\mathbf{ve_i}} \mathbf{ve_i} \exp(\sum\limits_{k=1}^{K} \theta_k f_k(\mathbf{ve_i}, y_i'))} \quad (2)$$

VEB learns a set weak learners $f_t$s iteratively and estimates the combined feature $F_t = F_{t-1} + f_t$ by solving the following weighted least square error(WLSE) problem:

$$f_t(\mathbf{ve}_i) = \arg\min_f \sum_{i=1}^{N} w_i E(f(\mathbf{ve}_i) - z_i)^2 = \arg\min_f [\sum_{i=1}^{N} \sum_{\mathbf{ve_i}} w_i p(y_i|\mathbf{ve_i})(f(\mathbf{ve}_i) - z_i)^2] \quad (3)$$

$$where \ w_i = p(y_i|\mathbf{ve_i})(1 - p(y_i|\mathbf{ve_i})), \quad z_i = \frac{y_i - 0.5}{p(y_i|\mathbf{ve_i})} \quad (4)$$

The $w_i$ and $z_i$ in equation 4 are the boosting weight and working response respectively for the $i^{th}$ data point, exactly as in LogitBoost. However, the least square problem for VEB (eq.3) involves $NX$ points because of virtual evidence as opposed to $N$ points in LogitBoost. Although eq. 4 is given for the binary case (i.e. $y_i \in \{0, 1\}$), it is easily extendible to the multi-class case and we have done that in our experiments. At each iteration, $\mathbf{ve}_i$ is updated as messages from $\mathbf{n}(y_i)$ changes with the addition of new features. We run belief propagation (BP) to obtain the virtual evidence before each iteration. The CRF feature weights, $\theta$'s are computed by solving the WLSE problem, where the local features, $n_{ki}$ is the count of feature $k$ in data instance $i$ and the compatibility features, $n_{ki}$ is the virtual evidence from the neighbors.: $\theta_k = \sum\limits_{i=1}^{N} w_i z_i n_{ki} / \sum\limits_{i=1}^{N} w_i n_{ki}$.

## 2.2 Semi-supervised training

For semi-supervised training of CRFs, Jiao *et.al.* [10] have proposed an algorithm that utilizes unlabeled data via entropy regularization – an extension of the approach proposed by [9] to structured CRF models. The objective function that is maximized during semi-supervised training of CRFs is given below, where $(\mathbf{x}_l, \mathbf{y}_l)$ and $(\mathbf{x}_u, \mathbf{y}_u)$ represent the labeled and unlabeled data respectively:

$$L_{SS}(\theta) = \log p(\mathbf{y}_l|\mathbf{x}_l, \theta) + \alpha \sum_{\mathbf{y}_u} p(\mathbf{y}_u|\mathbf{x}_u, \theta) \log p(\mathbf{y}_u|\mathbf{x}_u, \theta) - \|\theta\|/2$$

By minimizing the conditional entropy of the unlabeled data, the algorithm will generally find labeling of the unlabeled data that mutually reinforces the supervised labels. One drawback of this objective function is that it is no longer concave and in general there will be local maxima. The authors [10] showed that this method is still effective in improving an initial supervised model.

## 3 Semi-supervised virtual evidence boosting

In this work, we develop semi-supervised virtual evidence boosting (sVEB) that combines feature selection with semi-supervised training of CRFs. sVEB extends the VEB framework to take advantage of unlabeled data via minimum entropy regularization similar to [9, 10, 11]. The new objective function $L_{sVEB}$ we propose is as follows, where $(i = 1 \cdots N)$ are labeled and $(i = N + 1 \cdots M)$ are unlabeled examples:

$$L_{sVEB} = \sum_{i=1}^{N} \log p(y_i|\mathbf{ve_i}) + \alpha \sum_{i=N+1}^{M} \sum_{y_i'} p(y_i'|\mathbf{ve_i}) \log p(y_i'|\mathbf{ve_i}) \quad (5)$$

The sVEB aglorithm, similar to VEB, maximizes the conditional soft pseudo-likelihood of the labeled data but in addition minimizes the conditional entropy over unlabeled data. The $\alpha$ is a tuning parameter for controlling how much influence the unlabeled data will have.

By considering the soft pseudo-likelihood in $L_{sVEB}$ and using BP to estimate $p(y_i|\mathbf{ve_i})$, sVEB can use boosting to learn the parameters of CRFs. The virtual evidence from the neighboring nodes captures the label dependencies. There are three different types of feature functions $f$s that's used: for continuous observations $f_1(x_i)$ is a linear combination of decision stumps, for discrete observations the learner $f_2(x_i)$ is expressed as indicator functions, and for virtual evidences the weak learner $f_3(x_i)$ is the weighted sum of two indicator functions (for binary case). These functions are computed as follows, where $\delta$ is an indicator function, $h$ is a threshold for the decision stump, and $D$ is the number of dimensions of the observations:

$$f_1(x_i) = \theta_1\delta(x_i \geq h) + \theta_2\delta(x_i < h), \; f_2(x_i) = \sum_{k=1}^{D}\theta_k\delta(x_i = d), \; f_3(y_i) = \sum_{k=0}^{1}\theta_k\delta(y_i = k) \quad (6)$$

Similar to LogitBoost and VEB, the sVEB algorithm estimates a combined feature function $F$ that maximizes the objective by sequentially learning a set of weak learners, $f_t$'s (i.e. iteratively selecting features). In other words, sVEB solves the following weighted least-square error (WLSE) problem to learn $f_t$s:

$$f_t = \arg\min_f[\sum_{i=1}^{N}\sum_{\mathbf{ve}_i} w_i p(y_i|\mathbf{ve_i})(f(\mathbf{x_i}) - z_i)^2 + \sum_{i=N+1}^{M}\sum_{y_i'}\sum_{\mathbf{ve}_i} w_i p(y_i'|\mathbf{ve_i})(f(\mathbf{x_i}) - z_i)^2] \quad (7)$$

For labeled data (first term in eq.7), boosting weights, $w_i$'s, and working responses, $z_i$'s, are computed as described in equation 4. But for the case of unlabeled data the expression for $w_i$ and $z_i$ becomes more complicated because of the entropy term. We present the equations for $w_i$ and $z_i$ below, please refer to the Appendix for the derivations:

$$w_i = \alpha^2(1 - p(y_i|\mathbf{ve_i}))[p(y_i|\mathbf{v}e_i)(1 - p(y_i|\mathbf{ve_i})) + \log p(y_i|\mathbf{ve_i})]$$

$$z_i = \frac{(y_i - 0.5)p(y_i|\mathbf{ve_i})(1 - \log p(y_i|\mathbf{ve_i}))}{\alpha[p(y_i|\mathbf{ve_i})(1 - p(y_i|\mathbf{ve_i})) + \log p(y_i|\mathbf{ve_i})]} \quad (8)$$

The soft evidence corresponding to messages from the neighboring nodes is obtained by running BP on the entire training dataset (labeled and unlabeled). The CRF feature weights $\theta_k$s are computed by solving the WLSE problem (e.q.(7)), $\theta_k = \sum_{i=1}^{M}\sum_{y_i} w_i z_i n_{ki} / \sum_{i=1}^{M}\sum_{y_i} w_i n_{ki}$

Algorithm 1 gives the pseudo-code for sVEB. The main difference between VEB and sVEB are steps $7 - 10$, where we compute $w_i$'s and $z_i$'s for all possible values of $y_i$ based on the virtual evidence and observations of unlabeled training cases. The boosting weights and working responses are computed using equation (8). The weighted least-square error (WLSE) equation (eq. 7) in step 10 of sVEB is different from that of VEB and the solution results in slightly different CRF feature weights, $\theta$'s. One of the major advantages of VEB and sVEB over ML and sML is that the parameter estimation is done by mainly performing feature counting. Unlike ML and sML, we do not need to use an optimizer to learn the model parameters which results in a huge reduction in the time required to train the CRF models. Please refer to the complexity analysis section for details.

## 4 Experiments

We conduct two sets of experiments to evaluate the performance of the sVEB method for training CRFs and the advantage of performing feature selection as part of semi-supervised training. In the first set of experiments, we analyze how much the complexity of the underlying CRF and the tuning parameter $\alpha$ effect the performance using synthetic data. In the second set of experiments, we evaluate the benefit of feature selection and using unlabeled data on two real-world activity datasets.

We compare the performance of the semi-supervised virtual evidence boosting(sVEB) presented in this paper to the semi-supervised maximum likelihood (sML) method [10]. In addition, for the activity datasets, we also evaluate an alternative approach (sML+Boost), where a subset of features is selected in advance using boosting. To benchmark the performance of the semi-supervised techniques, we also evaluate three different supervised training approaches, namely maximum likelihood

| Algorithm 1: Training CRFs using semi-supervised VEB |
| --- |

**inputs** : structure of CRF and training data $(\mathbf{x}_i, y_i)$, with $y_i \in \{0, 1\}$, $1 \le i \le M$, and $F_0 = 0$
**output**: Learned $F_T$ and their corresponding weights, $\theta$

**1**  **for** $t = 1, 2, \cdots, T$ **do**
**2**      Run BP using $F_t$ to get virtual evidences $\mathbf{ve_i}$;
**3**      **for** $i = 1, 2, \cdots, N$ **do**
**4**          Compute likelihood $p(y_i | \mathbf{ve_i})$;
**5**          Compute $w_i$ and $z_i$ using equation (4)
**6**      **end**
**7**      **for** $i = N + 1, ..., M$ *and* $y_i = 0, 1$ **do**
**8**          Compute likelihood $p(y_i | \mathbf{ve_i})$;
**9**          Compute $w_i$ and $z_i$ using equation (8)
**10**     **end**
**11**     Obtain "best" weak learner $f_t$ according to equation (7) and update $F_t = F_{t-1} + f_t$ ;
**12** **end**

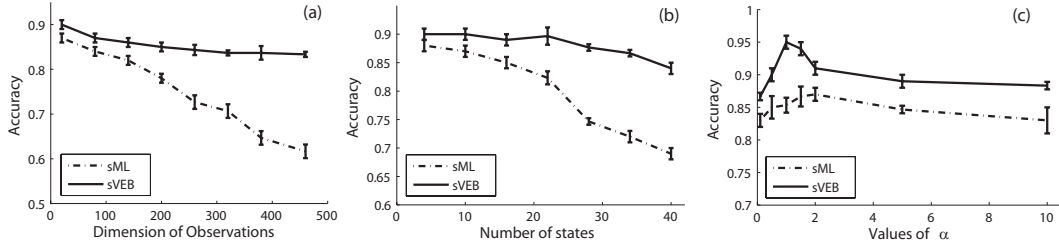

Figure 1: Accuracy of sML and sVEB for different number of states, local features and different values of $\alpha$.

method using all observed features(ML), (ML+Boost) using a subset of features selected in advance, and virtual evidence boosting (VEB). All the learned models are tested using standard maximum a posteriori(MAP) estimate and belief propagation. We used a $l_2$-norm shrinkage prior as a regularizer for the ML and sML methods.

## 4.1 Synthetic data

The synthetic data is generated using a first-order Markov Chain with self-transition probabilities set to 0.9. For each model, we generate five sequences of length 4,000 and divide each trace into sequences of length 200. We randomly choose 50% of them as the labeled and the other 50% as un-labeled training data. We perform leave-one-out cross-validation and report the average accuracies.

To measure how the complexity of the CRFs affects the performance of the different semi-supervised methods, we vary the number of local features and the number of states. First, we compare the performance of sVEB and sML on CRFs with increasing the number of features. The number of states is set to 10 and the number of observation features is varied from 20 to 400 observations. Figure (1a) shows the average accuracy for the two semi-supervised training methods and their confidence intervals. The experimental results demonstrate that sVEB outperforms sML as we increase the dimension of observations (i.e. the number of local features). In the second experiment, we increase the number of classes and keep the dimension of observations fixed to 100. Figure (1b) demonstrates that sVEB again outperforms sML as we increase the number of states. Given the same amount of training data, sVEB is less likely to overfit because of the feature selection step. In both these experiments we set the value of tuning parameter, $\alpha$, to 1.5. To explore the effect of tuning parameter $\alpha$, we vary the value of $\alpha$ from 0.1 to 10 , while setting the number of states to 10 and the number of dimensions to 100. Figure (1c) shows that the performance of both sML and sVEB depends on the value of $\alpha$ but the accuracy decreases for large $\alpha$'s similar to the sML results presented in [10].

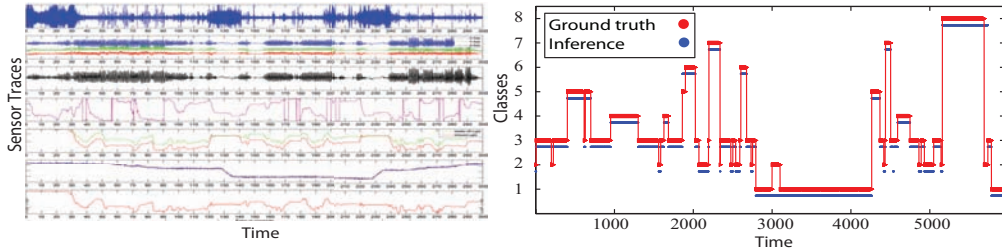

Figure 2: An example of a sensor trace and a classification trace

| Labeled | Average Accuracy (%) - Dataset 1 | | |
| --- | --- | --- | --- |
| | ML+all obs | ML+Boost | VEB |
| 60% | $62.7 \pm 6.6$ | $69.4 \pm 3.9$ | $82.6 \pm 7.3$ |
| 80% | $73.0 \pm 4.2$ | $81.8 \pm 4.7$ | $90.3 \pm 4.7$ |
| 100% | $77.8 \pm 3.4$ | $87.0 \pm 2.3$ | $91.5 \pm 3.8$ |

| Labeled | Average Accuracy (%) - Dataset 2 | | |
| --- | --- | --- | --- |
| | ML+all obs | ML+Boost | VEB |
| 60% | $74.3 \pm 3.7$ | $75.8 \pm 3.3$ | $88.5 \pm 5.1$ |
| 80% | $80.6 \pm 2.9$ | $84.8 \pm 2.9$ | $93.4 \pm 3.8$ |
| 100% | $86.2 \pm 3.1$ | $87.5 \pm 3.1$ | $93.8 \pm 4.6$ |

Table 1: Accuracy $\pm 95\%$ confidence interval of the supervised algorithms on activity datasets 1 and 2

## 4.2 Activity dataset

We collected two activity datasets using wearable sensors, which include audio, acceleration, light, temperature, pressure, and humidity. The first dataset contains instances of 8 basic physical activities (e.g. walking, running, going up/down stairs, going up/down elevator, sitting, standing, and brushing teeth) from 7 different users. There is on average 30 minutes of data per user and a total of 3.5 hours of data that is manually labeled for training and testing purposes. The data is segmented into $0.25s$ chunks resulting in a total of 49613 data points. For each chunk, we compute 651 features, which include signal energy in log and linear frequency bands, autocorrelation, different entropy measures, mean, variances *etc*. The features are chosen based on what is used in existing activity recognition literature and a few additional ones that we felt could be useful. During training, the data from each person is divided into sequences of length 200 and fed into linear chain CRFs as observations. The second dataset contains instances of 5 different indoor activities (e.g. computer usage, meal, meeting, watching TV and sleeping) from a single user. We recorded 15 hours of sensor traces over 12 days. As this set contains longer time-scale activities, the data is segmented into 1 minute chunks and 321 different features are computed, similar to the first dataset. There are a total of 907 data points. These features are fed into CRFs as observations, one linear chain CRF is created per day.

We evaluate the performance of supervised and semi-supervised training algorithms on these two datasets. For the semi-supervised case, we randomly select 40% of the sequences for a given person or a given day as labeled and a different subset as the unlabeled training data. We compare the performance of sML and sVEB as we incorporate more unlabeled data (20%, 40% and 60%) into the training process. We also compare the supervised techniques, ML, ML+Boost, and VEB, with increasing amount of labeled data. For all the experiments, the tuning parameter $\alpha$ is set to 1.5. We perform leave-one-person-out cross-validation on dataset 1 and leave-one-day-out cross-validation on dataset 2 and report the average the accuracies. The number of features chosen (i. e. through the boosting iterations) is set to 50 for both datasets – including more features did not significantly improve the classification performance.

For both datasets, incorporating more unlabeled data improves accuracy. The sML estimate of the CRF parameters performs the worst. Even with the shrinkage prior, the high dimensionality can still cause over-fitting and lower the accuracy. Whereas parameter estimation and feature selection via sVEB consistently results in the highest accuracy. The (sML+Boost) method performs better than sML but does not perform as well as when feature selection and parameter estimation is done within a unified framework as in sVEB. Table 2 summarize our results. The results of supervised learn-

| Un-labeled | Average Accuracy (%) - Dataset 1 | | |
| --- | --- | --- | --- |
| | sML+all obs | sML+Boost | sVEB |
| 20% | $60.8 \pm 5.4$ | $66.4 \pm 4.2$ | $72.6 \pm 2.3$ |
| 40% | $68.1 \pm 4.8$ | $76.8 \pm 3.4$ | $78.5 \pm 3.4$ |
| 60% | $74.9 \pm 3.1$ | $81.3 \pm 3.9$ | $85.3 \pm 4.1$ |

| Un-labeled | Average Accuracy (%) - Dataset 2 | | |
| --- | --- | --- | --- |
| | sML+all obs | sML+Boost | sVEB |
| 20% | $71.4 \pm 3.2$ | $70.5 \pm 5.3$ | $79.9 \pm 4.2$ |
| 40% | $73.5 \pm 5.8$ | $74.1 \pm 4.6$ | $83.5 \pm 6.3$ |
| 60% | $75.6 \pm 3.9$ | $77.8 \pm 3.2$ | $87.4 \pm 4.7$ |

Table 2: Accuracy $\pm 95\%$ confidence interval of semi-supervised algorithms on activity datasets 1 and 2

| Labeled | Average Accuracy (%) - Dataset 2 | | | Labeled | Average Accuracy (%) - Dataset 2 | | |
|---|---|---|---|---|---|---|---|
| | ML+all obs | ML+Boost | VEB | | ML+all obs | ML+Boost | VEB |
| 5% | $59.2 \pm 6.5$ | $65.7 \pm 8.3$ | $71.2 \pm 5.7$ | 5% | $71.2 \pm 4.1$ | $68.3 \pm 6.7$ | $79.7 \pm 7.9$ |
| 20% | $66.9 \pm 5.9$ | $67.3 \pm 8.5$ | $77.4 \pm 3.6$ | 20% | $71.4 \pm 6.3$ | $73.8 \pm 5.2$ | $83.1 \pm 6.4$ |

Table 3: Accuracy $\pm\ 95\%$ confidence interval of semi-supervised algorithms on activity datasets 1 and 2

ing algorithms are presented in Table 1. Similar to the semi-supervised results, the VEB method performs the best, the ML is the worst performer, and the accuracy numbers for the (ML+Boost) method is in between. The accuracy increases if we incorporate more labeled data during training. To evaluate sVEB when a small amount of labeled data is available, we performed another set of experiments on datasets 1 and 2, where only $5\%$ and $20\%$ of the training data is labeled respectively. We used all the available unlabeled data during training. The results are shown in table 3. These experiments clearly demonstrate that although adding more unlabeled data is not as helpful as incorporating more labeled data, the use of cheap unlabeled data along with feature selection can significantly boost the performance of the models.

### 4.3 Complexity Analysis

The sVEB and VEB algorithm are significantly faster than ML and sML because they do not need to use optimizers such as quasi-newton methods to learn the weight parameters. For each training iteration in sML the cost of running BP is $O(c_l ns^2 + c_u n^2 s^3)$ [10] whereas the cost of each boosting iteration in sVEB is $O((c_l + c_u)ns^2)$. An efficient entropy gradient computation is proposed in [17], which reduces the cost of sML to $O((c_l + c_u)ns^2)$ but still requires an optimizer to maximize the log-likelihood. Moreover, the number of training iterations needed is usually much higher than the number of boosting iterations because optimizers such as L-BFGS require many more iterations to reach convergence in high dimensional spaces. For example, for dataset 1, we needed about 1000 iterations for sML to converge but we ran sVEB for only 50 iterations. Table 4 shows the time for performing the experiments on activity datasets (as described in the previous section) [2]. On the other hand the space complexity of sVEB is linearly smaller than sML and ML. Similar to ML, sML has the space complexity of $O(ns^2 D)$ in the best case [10]. VEB and sVEB have a lower space cost of $O(ns^2 D_b)$, because of the feature selection step $D_b \ll D$ usually. Therefore, the difference becomes significant when we are dealing with high dimensional data, particularly if they include a large number of redundant features.

| | Time (hours) | | | | | | | | |
|---|---|---|---|---|---|---|---|---|---|
| | ML | ML+Boost | VEB | sML | sML+Boost | sVEB | | | |
| Dataset 1 | 34 | 18 | 2.5 | 96 | 48 | 4 | | | |
| Dataset 2 | 7.5 | 4.25 | 0.4 | 10.5 | 8 | 0.6 | | | |

| | |
|---|---|
| $n$ | length of training sequence |
| $c_l$ | number of labeled training sequences |
| $c_u$ | number of unlabeled training sequences |
| $s$ | number of states |
| $D, D_b$ | dimension of observations |

Table 4: Training time for the different algorithms.

## 5 Conclusion

We presented sVEB, a new semi-supervised training method for CRFs, that can simultaneously select discriminative features via modified LogitBoost and utilize unlabeled data via minimum-entropy regularization. Our experimental results demonstrate the sVEB significantly outperforms other training techniques in real-world activity recognition problems. The unified framework for feature selection and semi-supervised training presented in this paper reduces the computational and human labeling costs, which are often the major bottlenecks in building large classification systems.

### Acknowledgments
The authors would like to thank Nando de Freitas and Lin Liao for many helpful discussions. This work was supported by the NSF under grant number IIS 0433637 and NSERC Canada Graduate Scholarship.

## Footnotes

[1]When a prior is used in the maximum likelihood objective function as a regularizer – the second term in eq. (1), the method is in fact called maximum a posteriori.

## References

[1] J. Lafferty, A. McCallum, and F. Pereira. Conditional random fields: Probabilistic models for segmenting and labeling sequence data. In *Proc. of the International Conference on Machine Learning (ICML)*, 2001.

---

[2] The experiments were run in Matlab environment and as a result they took longer.

[2] Andrew McCallum. Efficiently inducing features or conditional random fields. In *Proc. of the Conference on Uncertainty in Artificial Intelligence (UAI)*, 2003.

[3] T. Dietterich, A. Ashenfelter, and Y. Bulatov. Training conditional random fields via gradient tree boosting. In *Proc. of the International Conference on Machine Learning (ICML)*, 2004.

[4] A. Torralba, K. P. Murphy, and W. T. Freeman. Contextual models for object detection using boosted random fields. In *Advances in Neural Information Processing Systems (NIPS)*, 2004.

[5] L. Liao, T. Choudhury, D. Fox, and H Kautz. Training conditional random fields using virtual evidence boosting. In *Proc. of the International Joint Conference on Artificial Intelligence (IJCAI)*, 2007.

[6] K. Nigam, A. McCallum, A. Thrun, and T. Mitchell. Text classification from labeled and unlabeled documents using em. *Machine learning*, 2000.

[7] A. Zhu, Z. Ghahramani, and J. Lafferty. Semi-supervised learning using gaussian fields and harmonic functions. In *Proc. of the International Conference on Machine Learning (ICML)*, 2003.

[8] W. Li and M. Andrew. Semi-supervised sequence modeling with syntactic topic models. In *Proc. of the National Conference on Artificial Intelligence (AAAI)*, 2005.

[9] Y. Grandvalet and Y. Bengio. Semi-supervised learning by entropy minimization. In *Advances in Neural Information Processing Systems (NIPS)*, 2004.

[10] F. Jiao, W. Wang, C. H. Lee, R. Greiner, and D. Schuurmans. Semi-supervised conditional random fields for improved sequence segmentation and labeling. In *International Committee on Computational Linguistics and the Association for Computational Linguistics*, 2006.

[11] C. Lee, S. Wang, F. Jiao, Schuurmans D., and R. Greiner. Learning to Model Spatial Dependency: Semi-Supervised Discriminative Random Fields. In *NIPS*, 2006.

[12] J.S. Yedidia, W.T. Freeman, and Y. Weiss. Constructing free-energy approximations and generalized belief propagation algorithms. *IEEE Transactions on Information Theory*, 51(7):2282–2312, 2005.

[13] Y. Weiss. Comparing mean field method and belief propagation for approximate inference in mrfs. 2001.

[14] J. Besag. Statistical analysis of non-lattice data. *The Statistician*, 24, 1975.

[15] C. J. Geyer and E. A. Thompson. Constrained Monte Carlo Maximum Likelihood for dependent data. *Journal of Royal Statistical Society*, 1992.

[16] Jerome Friedman, Trevor Hastie, and Robert Tibshirani. Additive logistic regression: a statistical view of boosting. *The Annals of Statistics*, 38(2):337–374, 2000.

[17] G. Mann and A. McCullum. Efficient computation of entropy gradient for semi-supervised conditional random fields. In *Human Language Technologies*, 2007.

# 6   Appendix

In this section, we show how we derived the equations for $w_i$ and $z_i$ (eq. 8):

$$L_F = L_{sVEB} = L_{VEB} - \alpha H_{emp} = \sum_{i=1}^{N} \log p(y_i|\mathbf{ve_i}) + \alpha \sum_{i=N+1}^{M} \sum_{y_i'} p(y_i'|\mathbf{ve_i}) \log p(y_i'|\mathbf{ve_i})$$

As in LogitBoost, the likelihood function $L_F$ is maximized by learning an ensemble of weak learners. We start with an empty ensemble $F = 0$ and iteratively add the next best weak learner, $f_t$, by computing the Newton update $\frac{s}{H}$, where $s$ and $H$ are the first and second derivative respectively of $L_F$ with respect to $f(\mathbf{ve_i}, y_i)$.

$$F(\mathbf{ve_i}, y_i)) \leftarrow F(\mathbf{ve_i}, y_i) - \frac{s}{H}, \text{ where } s = \frac{\partial L_{F+f}}{\partial f}|_{f=0} \text{ and } H = \frac{\partial^2 L_{F+f}}{\partial f^2}|_{f=0}$$

$$s = \sum_{i=1}^{N} 2(2y_i - 1)(1 - p(y_i|\mathbf{ve_i})) + \alpha \sum_{i=N+1}^{M} \sum_{y_i'} [2(2y_i' - 1)(1 - p(y_i'|\mathbf{ve_i}))p(y_i'|\mathbf{ve_i})(1 - \log p(y_i'|\mathbf{ve_i}))]$$

$$H = -\sum_{i=1}^{N} 4p(y_i|\mathbf{ve_i})(1 - p(y_i|\mathbf{ve_i}))(2y_i - 1)^2 + \alpha^2 \sum_{i=N+1}^{M} \sum_{y_i'} 4(2y_i' - 1)^2 (1 - p(y_i'|\mathbf{ve_i}))[p(y_i'|\mathbf{ve_i})(1 - p(y_i'|\mathbf{ve_i})) + \log p(y_i'|\mathbf{ve_i})]$$

$$F \leftarrow F + \frac{\sum_{i=1}^{N} z_i w_i + \sum_{i=N+1}^{M} \sum_{y_i'} z_i w_i}{\sum_{i=1}^{N} w_i + \sum_{i=N+1}^{M} \sum_{y_i'} w_i} \text{ where } z_i = \begin{cases} \frac{y_i - 0.5}{p(y_i|\mathbf{ve_i})} & \text{if } 1 \leq i \leq N \quad \text{eq. (4)} \\ \frac{(y_i' - 0.5)p(y_i'|\mathbf{ve_i})(1 - \log p(y_i'|\mathbf{ve_i}))}{\alpha[p(y_i'|\mathbf{ve_i})(1 - p(y_i'|\mathbf{ve_i})) + \log p(y_i'|\mathbf{ve_i})]} & \text{if } N < i \leq M \quad \text{eq. (8)} \end{cases}$$

$$\text{and} \quad w_i = \begin{cases} p(y_i|\mathbf{ve_i})(1 - p(y_i|\mathbf{ve_i})) & \text{if } 1 \leq i \leq N \quad \text{eq. (4)} \\ \alpha^2 (1 - p(y_i'|\mathbf{ve_i}))[p(y_i'|\mathbf{ve_i})(1 - p(y_i'|\mathbf{ve_i})) + \log p(y_i'|\mathbf{ve_i})] & \text{if } N < i \leq M \quad \text{eq. (8)} \end{cases}$$

At iteration $t$ we get the best weak learner, $f_t$, by solving the WLSE problem in eq. 7.

